# Dimensionality Reduction
# Using the Sparse Linear Model

**Ioannis Gkioulekas**
Harvard SEAS
Cambridge, MA 02138
igkiou@seas.harvard.edu

**Todd Zickler**
Harvard SEAS
Cambridge, MA 02138
zickler@seas.harvard.edu

## Abstract

We propose an approach for linear unsupervised dimensionality reduction, based on the sparse linear model that has been used to probabilistically interpret sparse coding. We formulate an optimization problem for learning a linear projection from the original signal domain to a lower-dimensional one in a way that approximately preserves, in expectation, pairwise inner products in the sparse domain. We derive solutions to the problem, present nonlinear extensions, and discuss relations to compressed sensing. Our experiments using facial images, texture patches, and images of object categories suggest that the approach can improve our ability to recover meaningful structure in many classes of signals.

## 1   Introduction

Dimensionality reduction methods are important for data analysis and processing, with their use motivated mainly from two considerations: (1) the impracticality of working with high-dimensional spaces along with the deterioration of performance due to the curse of dimensionality; and (2) the realization that many classes of signals reside on manifolds of much lower dimension than that of their ambient space. Linear methods in particular are a useful sub-class, for both the reasons mentioned above, and their potential utility in resource-constrained applications like low-power sensing [1, 2]. Principal component analysis (PCA) [3], locality preserving projections (LPP) [4], and neighborhood preserving embedding (NPE) [5] are some common approaches. They seek to reveal underlying structure using the global geometry, local distances, and local linear structure, respectively, of the signals in their original domain; and have been extended in many ways [6–8].[1]

On the other hand, it is commonly observed that geometric relations between signals in their original domain are only weakly linked to useful underlying structure. To deal with this, various feature transforms have been proposed to map signals to different (typically higher-dimensional) domains, with the hope that geometric relations in these alternative domains will reveal additional structure, for example by distinguishing image variations due to changes in pose, illumination, object class, and so on. These ideas have been incorporated into methods for dimensionality reduction by first mapping the input signals to an alternative (higher-dimensional) domain and then performing dimensionality reduction there, for example by treating signals as tensors instead of vectors [9, 10] or using kernels [11]. In the latter case, however, it can be difficult to design a kernel that is beneficial for a particular signal class, and ad hoc selections are not always appropriate.

In this paper, we also address dimensionality reduction through an intermediate higher-dimensional space: we consider the case in which input signals are samples from an underlying dictionary model. This generative model naturally suggests using the hidden covariate vectors as intermediate features, and learning a linear projection (of the original domain) to approximately preserve the Euclidean geometry of these vectors. Throughout the paper, we emphasize a particular instance of this model that is related to sparse coding, motivated by studies suggesting that data-adaptive sparse representations

are appropriate for signals such as natural images and facial images [12, 13], and enable state-of-the-art performance for denoising, deblurring, and classification tasks [14–19].

Formally, we assume our input signal to be well-represented by a *sparse linear model* [20], previously used for probabilistic sparse coding. Based on this generative model, we formulate learning a linear projection as an optimization problem with the objective of preservation, in expectation, of pairwise inner products between sparse codes, without having to explicitly obtain the sparse representation for each new sample. We study the solutions of this optimization problem, and we discuss how they are related to techniques proposed for compressed sensing. We discuss applicability of our results to general dictionary models, and nonlinear extensions. Finally, by applying our method to the visualization, clustering, and classification of facial images, texture patches, and general images, we show experimentally that it improves our ability to uncover useful structure. Omitted proofs and additional results can be found in the accompanying supplementary material.

## 2 The sparse linear model

We use $\mathbb{R}^N$ to denote the ambient space of the input signals, and assume that each signal $\boldsymbol{x} \in \mathbb{R}^N$ is generated as the sum of a noise term $\boldsymbol{\varepsilon} \in \mathbb{R}^N$ and a linear combination of the columns, or *atoms*, of a $N \times K$ *dictionary* matrix $\boldsymbol{D} = [\boldsymbol{d}_1, \ldots, \boldsymbol{d}_K]$, with the coefficients arranged as a vector $\boldsymbol{a} \in \mathbb{R}^K$,

$$\boldsymbol{x} = \boldsymbol{D}\boldsymbol{a} + \boldsymbol{\varepsilon}. \tag{1}$$

We assume the noise to be white Gaussian, $\boldsymbol{\varepsilon} \sim \mathcal{N}(\boldsymbol{0}_{N \times 1}, \sigma^2 \boldsymbol{I}_{N \times N})$. We are interested in the *sparse linear model* [20], according to which the elements of $\boldsymbol{a}$ are a-priori independent from $\boldsymbol{\varepsilon}$ and are identically and independently drawn from a Laplace distribution,

$$p(\boldsymbol{a}) = \prod_{i=1}^{K} p(a_i), \, p(a_i) = \frac{1}{2\tau} \exp\left\{-\frac{|a_i|}{\tau}\right\}. \tag{2}$$

In the context of this model, $\boldsymbol{D}$ is usually overcomplete ($K > N$), and in practice often learned in an unsupervised manner from training data. Several efficient algorithms exist for dictionary learning [21–23], and we assume in our analysis that a dictionary $\boldsymbol{D}$ adapted to the signals of interest is given.

Our adoption of the sparse linear model is motivated by significant empirical evidence that it is accurate for certain signals of interest, such as natural and facial images [12, 13], as well as the fact that it enables high performance for such diverse tasks as denoising and inpainting [14, 24], deblurring [15], and classification and clustering [13, 16–19]. Typically, the model (1) with an appropriate dictionary $\boldsymbol{D}$ is employed as a means for feature extraction, in which input signals $\boldsymbol{x}$ in $\mathbb{R}^N$ are mapped to higher-dimensional feature vectors $\boldsymbol{a} \in \mathbb{R}^K$. When inferring features $\boldsymbol{a}$ (termed *sparse codes*) through maximum-a-posteriori (MAP) estimation, they are solutions to

$$\min_{\boldsymbol{a}} \frac{1}{\sigma^2} \|\boldsymbol{x} - \boldsymbol{D}\boldsymbol{a}\|_2^2 + \frac{1}{\tau} \|\boldsymbol{a}\|_1. \tag{3}$$

This problem, known as the *lasso* [25], is a convex relaxation of the more general problem of *sparse coding* [26] (in the rest of the paper we use both terms interchangeably). A number of efficient algorithms for computing $\boldsymbol{a}$ exist, with both MAP [21, 27] and fully Bayesian [20] procedures.

## 3 Preserving inner products

Linear dimensionality reduction from $\mathbb{R}^N$ to $\mathbb{R}^M$, $M < N$, is completely specified by a projection matrix $\boldsymbol{L}$ that maps each $\boldsymbol{x} \in \mathbb{R}^N$ to $\boldsymbol{y} = \boldsymbol{L}\boldsymbol{x}$, $\boldsymbol{y} \in \mathbb{R}^M$, and different algorithms for linear dimensionality reduction correspond to different methods for finding this matrix. Typically, we are interested in projections that reveal useful structure in a given set of input signals.

As mentioned in the introduction, structure is often better revealed in a *higher-dimensional* space of features, say $\boldsymbol{a} \in \mathbb{R}^K$. When a suitable feature transform can be found, this structure may exist as simple Euclidean geometry and be encoded in pairwise Euclidean distances or inner products between feature vectors. This is used, for example, in support vector machines and nearest-neighbor classifiers based on Euclidean distance, as well as $k$-means and spectral clustering based on pairwise inner products. For the problem of dimensionality reduction, this motivates learning a projection matrix $\boldsymbol{L}$ such that, for any two input samples, the inner product between their resulting low-dimensional representations is close to that of their corresponding high-dimensional features.

More formally, for two samples $\boldsymbol{x}_k$, $k = 1, 2$ with corresponding low-dimensional representations $\boldsymbol{y}_k = \boldsymbol{L}\boldsymbol{x}_k$ and feature vectors $\boldsymbol{a}_k$, we define $\delta p = \boldsymbol{y}_1^T \boldsymbol{y}_2 - \boldsymbol{a}_1^T \boldsymbol{a}_2$ as a quantity whose magnitude we want *on average* to be small. Assuming that an accurate probabilistic generative model for the samples $\boldsymbol{x}$ and features $\boldsymbol{a}$ is available, we propose learning $\boldsymbol{L}$ by solving the optimization problem ($\mathbb{E}$ denoting expectation with respect to subscripted variables)

$$\min_{\boldsymbol{L}_{M \times N}} \mathbb{E}_{\boldsymbol{x}_1, \boldsymbol{x}_2, \boldsymbol{a}_1, \boldsymbol{a}_2} \left[ \delta p^2 \right]. \tag{4}$$

Solving (4) may in general be a hard optimization problem, depending on the model used for $\boldsymbol{a}_k$ and $\boldsymbol{x}_k$. Here we solve it for the case of the sparse linear model of Section 2, under which the feature vectors are the sparse codes. Using (1) and denoting $\boldsymbol{S} = \boldsymbol{L}^T \boldsymbol{L}$, (4) becomes

$$\min_{\boldsymbol{L}_{M \times N}} \mathbb{E}_{\boldsymbol{a}_1, \boldsymbol{a}_2, \boldsymbol{\varepsilon}_1, \boldsymbol{\varepsilon}_2} \left[ \left( \boldsymbol{a}_1^T \left( \boldsymbol{D}^T \boldsymbol{S} \boldsymbol{D} - \boldsymbol{I} \right) \boldsymbol{a}_2 + \boldsymbol{\varepsilon}_1^T \boldsymbol{S} \boldsymbol{D} \boldsymbol{a}_2 + \boldsymbol{\varepsilon}_2^T \boldsymbol{S} \boldsymbol{D} \boldsymbol{a}_1 + \boldsymbol{\varepsilon}_1^T \boldsymbol{S} \boldsymbol{\varepsilon}_2 \right)^2 \right]. \tag{5}$$

Assuming that $\boldsymbol{x}_1$ and $\boldsymbol{x}_2$ are drawn independently, we prove that (5) is equivalent to problem

$$\min_{\boldsymbol{L}_{M \times N}} 4\tau^4 \left\| \boldsymbol{D}^T \boldsymbol{S} \boldsymbol{D} - \boldsymbol{I} \right\|_F^2 + 4\tau^2 \sigma^2 \left\| \boldsymbol{S} \boldsymbol{D} \right\|_F^2 + \sigma^4 \left\| \boldsymbol{S} \right\|_F^2, \tag{6}$$

where $\|\cdot\|_F$ is the Frobenius norm, which has the closed-form solution (up to an arbitrary rotation):

$$\boldsymbol{L} = \operatorname{diag} \left( f \left( \boldsymbol{\lambda}_M \right) \right) \boldsymbol{V}_M^T. \tag{7}$$

Here, $\boldsymbol{\lambda}_M = (\lambda_1, \dots, \lambda_M)$ is a $M \times 1$ vector composed of the $M$ largest eigenvalues of the $N \times N$ matrix $\boldsymbol{D} \boldsymbol{D}^T$, and $\boldsymbol{V}_M$ is the $N \times M$ matrix with the corresponding eigenvectors as columns. The function $f(\cdot)$ is applied element-wise to the vector $\boldsymbol{\lambda}_M$ such that

$$f(\lambda_i) = \sqrt{\frac{4\tau^4 \lambda_i}{\sigma^4 + 4\tau^2 \sigma^2 \lambda_i + 4\tau^4 \lambda_i^2}}, \tag{8}$$

and $\operatorname{diag} \left( f \left( \boldsymbol{\lambda}_M \right) \right)$ is the $M \times M$ diagonal matrix formed from $f \left( \boldsymbol{\lambda}_M \right)$. This solution assumes that $\boldsymbol{D} \boldsymbol{D}^T$ has full rank $N$, which in practice is almost always true as $\boldsymbol{D}$ is overcomplete.

Through comparison with (5), we observe that (6) is a trade-off between bringing inner products of sparse codes and their projections close (first term), and suppressing noise (second and third terms). Their relative influence is controlled by the variance of $\boldsymbol{\varepsilon}$ and $\boldsymbol{a}$, through the constants $\sigma$ and $\tau$ respectively. It is interesting to compare their roles in (3) and (6): as $\sigma$ increases relative to $\tau$, data fitting in (3) becomes less important, and (7) emphasizes noise suppression. As $\tau$ increases, $l_1$-regularization in (3) is weighted less, and the first term in (6) more. In the extreme case of $\sigma = 0$, the data term in (3) becomes a hard constraint, whereas (6) and (7) simplify, respectively, to

$$\min_{\boldsymbol{L}_{M \times N}} \left\| \boldsymbol{D}^T \boldsymbol{S} \boldsymbol{D} - \boldsymbol{I} \right\|_F^2, \text{ and } \boldsymbol{L} = \operatorname{diag} \left( \boldsymbol{\lambda}_M \right)^{-\frac{1}{2}} \boldsymbol{V}_M^T. \tag{9}$$

Interestingly, in this noiseless case, an ambiguity arises in the solution of (9), as a minimizer is obtained for any subset of $M$ eigenpairs and not necessarily the $M$ largest ones.

The solution to (7) is similar—and in the noiseless case identical—to the whitening transform of the atoms of $\boldsymbol{D}$. When the atoms are centered at the origin, this essentially means that solving (4) for the sparse linear model amounts to *performing PCA on dictionary atoms learned from training samples instead of the training samples themselves*. The above result can also be interpreted in the setting of [28]: dimensionality reduction in the case of the sparse linear model with the objective of (4) corresponds to kernel PCA using the kernel $\boldsymbol{D} \boldsymbol{D}^T$, modulo centering and the normalization.

### 3.1 Other dictionary models

Even though we have presented our results using the sparse linear model described in Section 2, it is important to realize that our analysis is not limited to this model. The assumptions required for deriving (5) are that signals are generated by a linear dictionary model such as (1), where the coefficients of each of the noise and code vectors are independent and identically distributed according to some zero-mean distribution, with the two vectors also independent from each other. The above assumptions apply for several other popular dictionary models. Examples include the models used implicitly by ridge and bridge regression [29] and elastic-net [30], where the Laplace

prior on the code coefficients is replaced by a Gaussian, and priors of the form $\exp(-\lambda \|\boldsymbol{a}\|_q^q)$ and $\exp(-\lambda \|\boldsymbol{a}\|_1 - \gamma \|\boldsymbol{a}\|_2^2)$, respectively. In the context of sparse coding, other sparsity-inducing priors that have been proposed in the literature, such as Student's t-distribution [31], also fall into the same framework. We choose to emphasize the sparse linear model, however, due to the apparent structure present in dictionaries learned using this model, and its empirical success in diverse applications.

It is possible to derive similar results for a more general model. Specifically, we make the same assumptions as above, except that we only require that elements of $\boldsymbol{a}$ be zero-mean and not necessarily identically distributed, and similarly for $\varepsilon$. Then, we prove that (4) becomes

$$\min_{\boldsymbol{L}_{M \times N}} \left\| \left( \boldsymbol{D}^T \boldsymbol{S} \boldsymbol{D} - \boldsymbol{I} \right) \odot \sqrt{\boldsymbol{W}_1} \right\|_F^2 + \left\| (\boldsymbol{S} \boldsymbol{D}) \odot \sqrt{\boldsymbol{W}_2} \right\|_F^2 + \left\| \boldsymbol{S} \odot \sqrt{\boldsymbol{W}_3} \right\|_F^2, \quad (10)$$

where $\odot$ denotes the Hadamard product and $\left( \sqrt{\boldsymbol{W}} \right)_{ij} = \sqrt{(\boldsymbol{W})_{ij}}$. The elements of the weight matrices $\boldsymbol{W}_1$, $\boldsymbol{W}_2$ and $\boldsymbol{W}_3$ in (10), of sizes $K \times K$, $N \times K$, and $N \times N$ respectively, are

$$(\boldsymbol{W}_1)_{ij} = \mathbb{E}\left[a_{1i}^2 a_{2j}^2\right], \ (\boldsymbol{W}_2)_{ij} = \mathbb{E}\left[\varepsilon_{1i}^2 a_{2j}^2\right] + \mathbb{E}\left[\varepsilon_{2i}^2 a_{1j}^2\right], \ (\boldsymbol{W}_3)_{ij} = \mathbb{E}\left[\varepsilon_{1i}^2 \varepsilon_{2j}^2\right]. \quad (11)$$

Problem (10) can still be solved efficiently, see for example [32].

### 3.2 Extension to the nonlinear case

We consider a nonlinear extension of the above analysis through the use of kernels. We denote by $\Phi : \mathbb{R}^N \to \mathcal{H}$ a mapping from the signal domain to a reproducing kernel Hilbert space $\mathcal{H}$ associated with a kernel function $k : \mathbb{R}^N \times \mathbb{R}^N \to \mathbb{R}$ [33]. Using a set $\mathcal{D} = \{\tilde{d}_i \in \mathcal{H}, i = 1, \ldots, K\}$ as dictionary, we extend the sparse linear model of Section 2 by replacing (1) for each $\boldsymbol{x} \in \mathbb{R}^N$ with

$$\Phi(\boldsymbol{x}) = \mathcal{D}\boldsymbol{a} + \tilde{\varepsilon}, \quad (12)$$

where $\mathcal{D}\boldsymbol{a} \equiv \sum_{i=1}^K a_i \tilde{d}_i$. For $\boldsymbol{a} \in \mathbb{R}^K$ we make the same assumptions as in the sparse linear model. The term $\tilde{\varepsilon}$ denotes a *Gaussian process* over the domain $\mathbb{R}^N$ whose sample paths are functions in $\mathcal{H}$ and with *covariance operator* $C_{\tilde{\varepsilon}} = \sigma^2 \mathcal{I}$, where $\mathcal{I}$ is the identity operator on $\mathcal{H}$ [33, 34].

This nonlinear extension of the sparse linear model is valid only in finite dimensional spaces $\mathcal{H}$. In the infinite dimensional case, constructing a Gaussian process with both sample paths in $\mathcal{H}$ and identity covariance operator is not possible, as that would imply that the identity operator in $\mathcal{H}$ has finite Hilbert-Schmidt norm [33, 34]. Related problems arise in the construction of cylindrical Gaussian measures on infinite dimensional spaces [35]. We define $\tilde{\varepsilon}$ this way to obtain a probabilistic model for which MAP inference of $\boldsymbol{a}$ corresponds to the kernel extension of the lasso (3) [36],

$$\min_{\boldsymbol{a} \in \mathbb{R}^K} \frac{1}{2\sigma^2} \|\Phi(\boldsymbol{x}) - \mathcal{D}\boldsymbol{a}\|_{\mathcal{H}}^2 + \frac{1}{\tau} \|\boldsymbol{a}\|_1, \quad (13)$$

where $\|\cdot\|_{\mathcal{H}}$ is the norm $\mathcal{H}$ defined through $k$. In the supplementary material, we discuss an alternative to (12) that resolves these problems by requiring that all $\Phi(\boldsymbol{x})$ be in the subspace spanned by the atoms of $\mathcal{D}$. Our results can be extended to this alternative, however in the following we adopt (12) and limit ourselves to finite dimensional spaces $\mathcal{H}$, unless mentioned otherwise.

In the kernel case, the equivalent of the projection matrix $\boldsymbol{L}$ (transposed) is a compact, linear operator $\mathcal{V} : \mathcal{H} \to \mathbb{R}^M$, that maps an element $\boldsymbol{x} \in \mathbb{R}^N$ to $\boldsymbol{y} = \mathcal{V}\Phi(\boldsymbol{x}) \in \mathbb{R}^M$. We denote by $\mathcal{V}^* : \mathbb{R}^M \to \mathcal{H}$ the adjoint of $\mathcal{V}$, and by $\mathcal{S} : \mathcal{H} \to \mathcal{H}$ the self-adjoint positive semi-definite linear operator of rank $M$ from their synthesis, $\mathcal{S} = \mathcal{V}^* \mathcal{V}$. If we consider optimizing over $\mathcal{S}$, we prove that (4) reduces to

$$\min_{\mathcal{S}} 4\tau^4 \sum_{i=1}^K \sum_{i=1}^K \left( \left\langle \tilde{d}_i, \mathcal{S}\tilde{d}_j \right\rangle_{\mathcal{H}} - \delta_{ij} \right)^2 + 4\tau^2 \sigma^2 \sum_{i=1}^K \left\langle \mathcal{S}\tilde{d}_i, \mathcal{S}\tilde{d}_i \right\rangle_{\mathcal{H}} + \|\mathcal{S}\|_{HS}^2, \quad (14)$$

where $\|\cdot\|_{HS}$ is the Hilbert-Schmidt norm. Assuming that $\boldsymbol{K}_{\mathcal{D}\mathcal{D}}$ has full rank (which is almost always true in practice due to the very large dimension of the Hilbert spaces used) we extend the representer theorem of [37] to prove that all solutions of (14) can be written in the form

$$\mathcal{S} = (\mathcal{D}\boldsymbol{B}) \otimes (\mathcal{D}\boldsymbol{B}), \quad (15)$$

where $\otimes$ denotes the tensor product between all pairs of elements of its operands, and $\boldsymbol{B}$ is a $K \times M$ matrix. Then, denoting $\boldsymbol{Q} = \boldsymbol{B}\boldsymbol{B}^T$, problem (14) becomes

$$\min_{\boldsymbol{B}_{K \times M}} 4\tau^4 \left\| \boldsymbol{K}_{\mathcal{D}\mathcal{D}} \boldsymbol{Q} \boldsymbol{K}_{\mathcal{D}\mathcal{D}} - \boldsymbol{I} \right\|_F^2 + 4\tau^2 \sigma^2 \left\| \boldsymbol{K}_{\mathcal{D}\mathcal{D}} \boldsymbol{Q} \boldsymbol{K}_{\mathcal{D}\mathcal{D}}^{\frac{1}{2}} \right\|_F^2 + \sigma^4 \left\| \boldsymbol{K}_{\mathcal{D}\mathcal{D}}^{\frac{1}{2}} \boldsymbol{Q} \boldsymbol{K}_{\mathcal{D}\mathcal{D}}^{\frac{1}{2}} \right\|_F^2, \quad (16)$$

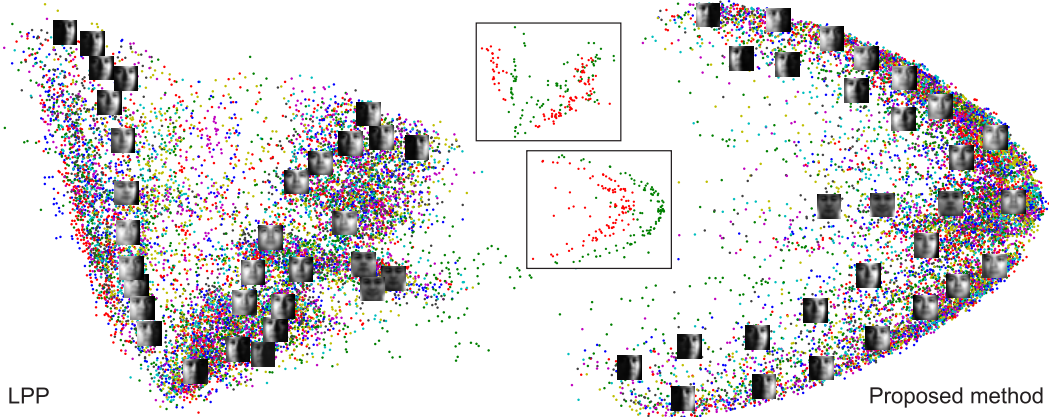

Figure 1: Two-dimensional projection of CMU PIE dataset, colored by identity. Shown at high resolution and at their respective projections are identity-averaged faces across the dataset for various illuminations, poses, and expressions. Insets show projections of samples from only two distinct identities. (Best viewed in color.)

where $\boldsymbol{K}_{\mathcal{DD}}(i,j) = \langle \tilde{d}_i, \tilde{d}_j \rangle_{\mathcal{H}}$, $i,j = 1, \ldots, K$. We can replace $\tilde{\boldsymbol{L}} = \boldsymbol{B}^T \boldsymbol{K}_{\mathcal{DD}}^{\frac{1}{2}}$ to turn (16) into an equivalent problem over $\tilde{\boldsymbol{L}}$ of the form (6), with $\boldsymbol{K}_{\mathcal{DD}}^{\frac{1}{2}}$ instead of $\boldsymbol{D}$, and thus use (8) to obtain

$$\boldsymbol{B} = \boldsymbol{V}_M \operatorname{diag}\left(g\left(\boldsymbol{\lambda}_M\right)\right) \tag{17}$$

where, similar to the linear case, $\boldsymbol{\lambda}_M$ and $\boldsymbol{V}_M$ are the $M$ largest eigenpairs of the matrix $\boldsymbol{K}_{\mathcal{DD}}$, and

$$g\left(\lambda_i\right) = \frac{1}{\sqrt{\lambda_i}} f\left(\lambda_i\right) = \sqrt{\frac{4\tau^4}{\sigma^4 + 4\tau^2\sigma^2\lambda_i + 4\tau^4\lambda_i^2}}. \tag{18}$$

Using the derived solution, a vector $\boldsymbol{x} \in R^N$ is mapped to $\boldsymbol{y} = \boldsymbol{B}^T \boldsymbol{K}_{\mathcal{D}}(\boldsymbol{x})$, where $\boldsymbol{K}_{\mathcal{D}}(\boldsymbol{x}) = [\langle \tilde{d}_1, \Phi(\boldsymbol{x}) \rangle_{\mathcal{H}}, \ldots, \langle \tilde{d}_M, \Phi(\boldsymbol{x}) \rangle_{\mathcal{H}}]^T$. As in the linear case, this is similar to the result of applying kernel PCA on the dictionary $\mathcal{D}$ instead of the training samples. Note that, in the noiseless case, $\sigma = 0$, the above analysis is also valid for infinite dimensional spaces $\mathcal{H}$. Expression (17) simplifies to $\boldsymbol{B} = \boldsymbol{V}_M \operatorname{diag}\left(\boldsymbol{\lambda}_M\right)^{-1}$ where, as in the linear case, any subset of $M$ eigenvalues may be selected. Even though in the infinite dimensional case selecting the $M$ largest eigenvalues cannot be justified probabilistically, it is a reasonable heuristic given the analysis in the finite dimensional case.

### 3.3 Computational considerations

It is interesting to compare the proposed method in the nonlinear case with kernel PCA, in terms of computational and memory requirements. If we require dictionary atoms to have pre-images in $\mathbb{R}^N$, that is $\mathcal{D} = \left\{ \Phi\left(d_i\right), d_i \in \mathbb{R}^N, i = 1, \ldots, K \right\}$ [36], then the proposed algorithm requires calculating and decomposing the $K \times K$ kernel matrix $\boldsymbol{K}_{\mathcal{DD}}$ when learning $\mathcal{V}$, and performing $K$ kernel evaluations for projecting a new sample $\boldsymbol{x}$. For kernel PCA on the other hand, the $S \times S$ matrix $\boldsymbol{K}_{\mathcal{XX}}$ and $S$ kernel evaluations are needed respectively, where $\mathcal{X} = \left\{ \Phi\left(x_i\right), x_i \in \mathbb{R}^N, i = 1, \ldots, S \right\}$ and $x_i$ are the representations of the training samples in $\mathcal{H}$, with $S \gg K$. If the pre-image constraint is dropped and the usual alternating procedure [21] is used for learning $\mathcal{D}$, then the representer theorem of [38] implies that $\mathcal{D} = \mathcal{X}\boldsymbol{F}$, where $\boldsymbol{F}$ is an $S \times K$ matrix. In this case, the proposed method also requires calculating $\boldsymbol{K}_{\mathcal{XX}}$ during learning and $S$ kernel evaluations for out-of-sample projections, but only the eigendecomposition of the $K \times K$ matrix $\boldsymbol{F}^T \boldsymbol{K}_{\mathcal{XX}}^2 \boldsymbol{F}$ is required.

On the other hand, we have assumed so far, in both the linear and nonlinear cases, that a dictionary is given. When this is not true, we need to take into account the cost of learning a dictionary, which greatly outweights the computational savings described above, despite advances in dictionary learning algorithms [21, 22]. In the kernel case, whereas imposing the pre-image constraint has the advantages we mentioned, it also makes dictionary learning a harder nonlinear optimization problem, due to the need for evaluation of kernel derivatives. In the linear case, the computational savings from applying (linear) PCA to the dictionary instead of the training samples are usually negligible, and therefore the difference in required computation becomes even more severe.

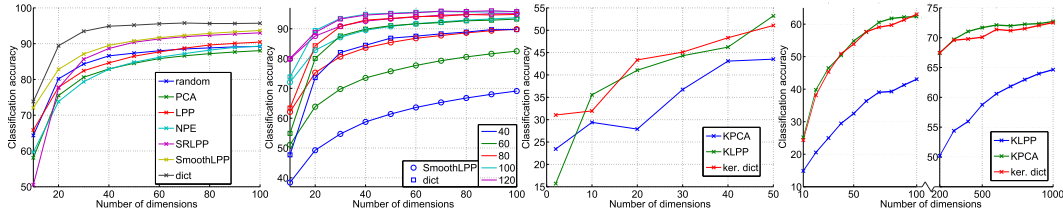

Figure 2: Classification accuracy results. From left to right: CMU PIE (varying value of $M$); CMU PIE (varying number of training samples); brodatz texture patches; Caltech-101. (Best viewed in color.)

## 4  Experimental validation

In order to evaluate our proposed method, we compare it with other unsupervised dimensionality reduction methods on visualization, clustering, and classification tasks. We use facial images in the linear case, and texture patches and images of object categories in the kernel case.

**Facial images:** We use the CMU PIE [39] benchmark dataset of faces under pose, illumination and expression changes, and specifically the subset used in [8].[2] We visualize the dataset by projecting all face samples to $M = 2$ dimensions using LPP and the proposed method, as shown in Figure 1. Also shown are identity-averaged faces over the dataset, for various illumination, pose, and expression combinations, at the location of their projection. We observe that our method recovers a very clear geometric structure, with changes in illumination corresponding to an ellipsoid, changes in pose to moving towards its interior, and changes in expression accounting for the density on the horizontal axis. We separately show the projections of samples from two distinct indviduals, and see that different identities are mapped to parallely shifted ellipsoids, easily separated by a nearest-neighbor classifier. On the other hand, such structure is not apparent when using LPP. A larger version of Figure 1 and the corresponding for PCA are provided in the supplementary material.

To assess how well identity structure is recovered for increasing values of the target dimension $M$, we also perform face recognition experiments. We compare against three baseline methods, PCA, NPE, and LPP, linear extensions (spectral regression "SRLPP" [7], spatially smooth LPP "SmoothLPP" [8]), and random projections (see Section 5). We produce 20 random splits into training and testing sets, learn a dictionary and projection matrices from the training set, and use the obtained low-dimensional representations with a $k$-nearest neighbor classifier ($k = 4$) to classify the test samples, as is common in the literature. In Figure 2, we show the average recognition accuracy for the various methods as the number of projections is varied, when using 100 training samples for each of the 68 individuals in the dataset. Also, we compare the proposed method with the best performing alternative, when the number of training samples per individual is varied from 40 to 120. We observe that the proposed method outperforms all other by a wide margin, in many cases even when trained with fewer samples. However, it can only be used when there are enough training samples to learn a dictionary, a limitation that does not apply to the other methods. For this reason, we do not experiment with cases of 5-20 samples per individual, as commonly done in the literature.

**Texture patches:** We perform classification experiments on texture patches, using the Brodatz dataset [40], and specifically classes 4, 5, 8, 12, 17, 84, and 92 from the 2-texture images. We extract $12 \times 12$ patches and use those from the training images to learn dictionaries and projections for the Gaussian kernel.[3] We classify the low-dimensional representations using an one-versus-all linear SVM. In Figure 2, we compare the classification accuracy of the proposed method ("ker.dict") with the kernel variants of PCA and LPP ("KPCA" and "KLPP" respectively), for varying $M$. KLPP and the proposed method both outperform KPCA. Our method achieves much higher accuracy at small values of $M$, and KLPP is better for large values; otherwise they perform similarly.

This dataset provides an illustrative example for the discussion in Section 3.3. For 20000 training samples, KPCA and KLPP require storing and processing a $20000 \times 20000$ kernel matrix, as opposed to $512 \times 512$ for our method. On the other hand, training a dictionary with $K = 512$ for this dataset takes approximately 2 hours, on an 8 core machine and using a C++ implementation of the learning algorithm, as opposed to the few minutes required for the eigendecompositions in KPCA and KLPP.

| Method | Accuracy | NMI | Rand Index |
|---|---|---|---|
| KPCA ($k$-means) | 0.6217 | 0.6380 | 0.4279 |
| KLPP (spectral clustering) | 0.6900 | 0.6788 | 0.5143 |
| ker.dict ($k$-means) | **0.7233** | **0.7188** | **0.5275** |

Table 1: Clustering results on Caltech-101.

**Images of object categories:** We use the Caltech-101 [41] object recognition dataset, with the average of the 39 kernels used in [42]. Firstly, we use 30 training samples from each class to learn a dictionary[4] and projections using KPCA, KLPP, and the proposed method. In Figure 2, we plot the classification accuracy achieved using a linear SVM for each method and varying $M$. We see that the proposed method and KPCA perform similarly and outperform KLPP. Our algorithm performs consistently well in both the datasets we experiment with in the kernel case.

We also perform unsupervised clustering experiments, where we randomly select 30 samples from each of the 20 classes used in [43] to learn projections with the three methods, over a range of values for $M$ between 10 and 150. We combine each with three clustering algorithms, $k$-means, spectral clustering [44], and affinity propagation [43] (using negative Euclidean distances of the low-dimensional representations as similarities). In Table 1, we report for each method the best overall result in terms of accuracy, normalized mutual information, and rand index [45], along with the clustering algorithm for which these are achieved. We observe that the low-dimensional representations from the proposed method produce the best quality clusterings, for all three measures.

## 5 Discussion and future directions

As we remarked in Section 3, the proposed method uses available training samples to learn $D$ and ignores them afterwards, relying exclusively on the assumed generative model and the correlation information in $D$. To see how this approach could fail, consider the degenerate case when $D$ is the identity matrix, that is the signal and sparse domains coincide. Then, to discover structure we need to directly examine the training samples. Better use of the training samples within our framework can be made by adopting a richer probabilistic model, using available data to train it, naturally with appropriate regularization to avoid overfitting, and then minimizing (4) for the learned model. For example, we can use the more general model of Section 3.1, and assume that each $a_i$ follows a Laplace distribution with a different $\tau_i$. Doing so agrees with empirical observations that, when $D$ is learned, the average magnitude of coefficients $a_i$ varies significantly with $i$. An orthogonal approach is to forgo adopting a generative model, and learn a projection matrix directly from training samples using an appropriate *empirical* loss function. One possibility is minimizing $\|A^T A - X^T L^T L X\|_F^2$, where the columns of $X$ and $A$ are the training samples and corresponding sparse code estimates, which is an instance of multidimensional scaling [46] (as modified to achieve linear induction).

For the sparse linear model case, objective function (4) is related to the *Restricted Isometry Property* (RIP) [47], used in the compressed sensing literature as a condition enabling reconstruction of a sparse vector $a \in \mathbb{R}^K$ from linear measurements $y \in \mathbb{R}^M$ when $M \ll K$. The RIP is a worst-case condition, requiring approximate preservation, in the low-dimensional domain, of pairwise Euclidean distances of all $a$, and therefore stronger than the expectation condition (4). Verifying the RIP for an arbitrary matrix is a hard problem, but it is known to hold for the *equivalent dictionary* $\tilde{D} = LD$ with high probability, if $L$ is drawn from certain random distributions, and $M$ is of the order of only $O\left(k \log \frac{K}{k}\right)$ [48]. Despite this property, our experiments demonstrate that a learned matrix $L$ is in practice more useful than random projections (see left of Figure 2). The formal guarantees that preservation of Euclidean geometry of sparse codes is possible with few linear projections are unique for the sparse linear model, thus further justifying our choice to emphasize this model throughout the paper.

Another quantity used in compressed sensing is the *mutual coherence* of $\tilde{D}$ [49], and its approximate minimization has been proposed as a way for learning $L$ for signal reconstruction [50, 51]. One of the optimization problems arrived at in this context [51] is the same as problem (9) we derived in the noiseless case, the solution of which as we mentioned in Section 3 is not unique. This ambiguity has been addressed heuristically by weighting the objective function with appropriate multiplicative terms, so that it becomes $\|\Lambda - \Lambda V^T L^T L V \Lambda\|_F^2$, where $\Lambda$ and $V$ are eigenpairs of $DD^T$ [51]. This

problem admits as only minimizer the one corresponding to the $M$ largest eigenvalues. Our analysis addresses the above issue naturally by incorporating noise, thus providing formal justification for the heuristic. Also, the closed-form solution of (9) is not shown in [51], though its existence is mentioned, and the (weighted) problem is instead solved through an iterative procedure.

In Section 3, we motivated preserving inner products in the sparse domain by considering existing algorithms that employ sparse codes. As our understanding of sparse coding continues to improve [52], there is motivation for considering other structure in $\mathbb{R}^K$. Possibilities include preservation of linear subspace (as determined by the support of the sparse codes) or local group relations in the sparse domain. Extending our analysis to also incorporate supervision is another important future direction.

Linear dimensionality reduction has traditionally been used for data preprocessing and visualization, but we are also beginning to see its utility for low-power sensors. A sensor can be designed to record linear projections of an input signal, instead of the signal itself, with projections implemented through a low-power physical process like optical filtering. In these cases, methods like the ones proposed in this paper can be used to obtain a small number of informative projections, thereby reducing the power and size of the sensor while maintaining its effectiveness for tasks like recognition. An example for visual sensing is described in [2], where a heuristically-modified version of our linear approach is employed to select projections for face detection. Rigorously extending our analysis to this domain will require accounting for noise and constraints on the projections (for example non-negativity, limited resolution) induced by fabrication processes. We view this as a research direction worth pursuing.

**Acknowledgments**

This research was supported by NSF award IIS-0926148, ONR award N000140911022, and the US Army Research Laboratory and the US Army Research Office under contract/grant number 54262-CI.

## Footnotes

[1]Other linear methods, most notably linear discriminant analysis (LDA), exploit class labels to learn projections. In this paper, we focus on the unsupervised setting.

[2]Images are pre-normalized to unit length. We use the algorithm of [21] to learn dictionaries, with $K$ equal to the number of pixels $N = 1024$, due to the limited amount of training data, and $\lambda = \frac{\sigma^2}{\tau} = 0.05$ as in [19].

[3]Following [36], we set the kernel parameter $\gamma = 8$, and use their method for dictionary learning with $K = 512$ and $\lambda = 0.30$, but with a conjugate gradient optimizer for the dictionary update step.

[4]We use a kernel extension of the algorithm of [21] without pre-image constraints. We select $K = 300$ and $\lambda = 0.1$ from a range of values, to achieve about 10% non-zero coefficients in the sparse codes and small reconstruction error for the training samples. Using $K = 150$ or $600$ affected accuracy by less than 1.5%.

# References

[1] M.A. Davenport, P.T. Boufounos, M.B. Wakin, and R.G. Baraniuk. Signal processing with compressive measurements. *IEEE JSTSP*, 2010.

[2] S.J. Koppal, I. Gkioulekas, T. Zickler, and G.L. Barrows. Wide-angle micro sensors for vision on a tight budget. *CVPR*, 2011.

[3] I. Jolliffe. *Principal component analysis*. Wiley, 1986.

[4] X. He and P. Niyogi. Locality Preserving Projections. *NIPS*, 2003.

[5] X. He, D. Cai, S. Yan, and H.J. Zhang. Neighborhood preserving embedding. *ICCV*, 2005.

[6] D. Cai, X. He, J. Han, and H.J. Zhang. Orthogonal laplacianfaces for face recognition. *IEEE IP*, 2006.

[7] D. Cai, X. He, and J. Han. Spectral regression for efficient regularized subspace learning. *ICCV*, 2007.

[8] D. Cai, X. He, Y. Hu, J. Han, and T. Huang. Learning a spatially smooth subspace for face recognition. *CVPR*, 2007.

[9] X. He, D. Cai, and P. Niyogi. Tensor subspace analysis. *NIPS*, 2006.

[10] J. Ye, R. Janardan, and Q. Li. Two-dimensional linear discriminant analysis. *NIPS*, 2004.

[11] B. Scholkopf, A. Smola, and K.R. Muller. Nonlinear component analysis as a kernel eigenvalue problem. *Neural computation*, 1998.

[12] B.A. Olshausen and D.J. Field. Sparse coding with an overcomplete basis set: A strategy employed by V1? *Vision Research*, 1997.

[13] J. Wright, A.Y. Yang, A. Ganesh, S.S. Sastry, and Y. Ma. Robust face recognition via sparse representation. *PAMI*, 2008.

[14] M. Elad and M. Aharon. Image denoising via sparse and redundant representations over learned dictionaries. *IEEE IP*, 2006.

[15] J.F. Cai, H. Ji, C. Liu, and Z. Shen. Blind motion deblurring from a single image using sparse approximation. *CVPR*, 2009.

[16] R. Raina, A. Battle, H. Lee, B. Packer, and A.Y. Ng. Self-taught learning: Transfer learning from unlabeled data. *ICML*, 2007.

[17] J. Mairal, F. Bach, J. Ponce, G. Sapiro, and A. Zisserman. Supervised dictionary learning. *NIPS*, 2008.

[18] I. Ramirez, P. Sprechmann, and G. Sapiro. Classification and clustering via dictionary learning with structured incoherence and shared features. *CVPR*, 2010.

[19] J. Yang, K. Yu, and T. Huang. Supervised translation-invariant sparse coding. *CVPR*, 2010.

[20] M.W. Seeger. Bayesian inference and optimal design for the sparse linear model. *JMLR*, 2008.

[21] H. Lee, A. Battle, R. Raina, and A.Y. Ng. Efficient sparse coding algorithms. *NIPS*, 2007.

[22] J. Mairal, F. Bach, J. Ponce, and G. Sapiro. Online learning for matrix factorization and sparse coding. *JMLR*, 2010.

[23] M. Zhou, H. Chen, J. Paisley, L. Ren, G. Sapiro, and L. Carin. Non-Parametric Bayesian Dictionary Learning for Sparse Image Representations. *NIPS*, 2009.

[24] J. Mairal, F. Bach, J. Ponce, G. Sapiro, and A. Zisserman. Non-local sparse models for image restoration. *ICCV*, 2009.

[25] R. Tibshirani. Regression shrinkage and selection via the lasso. *JRSS-B*, 1996.

[26] A.M. Bruckstein, D.L. Donoho, and M. Elad. From sparse solutions of systems of equations to sparse modeling of signals and images. *SIAM review*, 2009.

[27] B. Efron, T. Hastie, I. Johnstone, and R. Tibshirani. Least angle regression. *Annals of statistics*, 2004.

[28] J. Ham, D.D. Lee, S. Mika, and B. Schölkopf. A kernel view of the dimensionality reduction of manifolds. *ICML*, 2004.

[29] W.J. Fu. Penalized regressions: the bridge versus the lasso. *JCGS*, 1998.

[30] H. Zou and T. Hastie. Regularization and variable selection via the elastic net. *JRSS-B*, 2005.

[31] S. Ji, Y. Xue, and L. Carin. Bayesian compressive sensing. *IEEE SP*, 2008.

[32] N. Srebro and T. Jaakkola. Weighted low-rank approximations. *ICML*, 2003.

[33] A. Berlinet and C. Thomas-Agnan. *Reproducing kernel Hilbert spaces in probability and statistics*. Kluwer, 2004.

[34] V.I. Bogachev. *Gaussian measures*. AMS, 1998.

[35] J. Kuelbs, FM Larkin, and J.A. Williamson. Weak probability distributions on reproducing kernel hilbert spaces. *Rocky Mountain J. Math*, 1972.

[36] S. Gao, I. Tsang, and L.T. Chia. Kernel Sparse Representation for Image Classification and Face Recognition. *ECCV*, 2010.

[37] J. Abernethy, F. Bach, T. Evgeniou, and J.P. Vert. A new approach to collaborative filtering: Operator estimation with spectral regularization. *JMLR*, 2009.

[38] B. Scholkopf, R. Herbrich, and A. Smola. A generalized represeter theorem. *COLT*, 2001.

[39] T. Sim, S. Baker, and M. Bsat. The CMU pose, illumination, and expression (PIE) database. *IEEE ICAFGR*, 2002.

[40] T. Randen and J.H. Husoy. Filtering for texture classification: A comparative study. *PAMI*, 2002.

[41] L. Fei-Fei, R. Fergus, and P. Perona. Learning generative visual models from few training examples: an incremental bayesian approach tested on 101 object categories. *CVPR Workshops*, 2004.

[42] P. Gehler and S. Nowozin. On feature combination for multiclass object classification. *ICCV*, 2009.

[43] D. Dueck and B.J. Frey. Non-metric affinity propagation for unsupervised image categorization. *ICCV*, 2007.

[44] J. Shi and J. Malik. Normalized cuts and image segmentation. *PAMI*, 2000.

[45] N.X. Vinh, J. Epps, and J. Bailey. Information theoretic measures for clusterings comparison: Variants, properties, normalization and correction for chance. *JMLR*, 2010.

[46] T.F. Cox and M.A.A. Cox. *Multidimensional Scaling*. Chapman & Hall, 2000.

[47] E.J. Candès and T. Tao. Decoding by linear programming. *IEEE IT*, 2005.

[48] H. Rauhut, K. Schnass, and P. Vandergheynst. Compressed sensing and redundant dictionaries. *IEEE IT*, 2008.

[49] D.L. Donoho and X. Huo. Uncertainty principles and ideal atomic decomposition. *IEEE IT*, 2001.

[50] M. Elad. Optimized projections for compressed sensing. *IEEE SP*, 2007.

[51] J.M. Duarte-Carvajalino and G. Sapiro. Learning to sense sparse signals: Simultaneous sensing matrix and sparsifying dictionary optimization. *IEEE IP*, 2009.

[52] K. Yu, T. Zhang, and Y. Gong. Nonlinear learning using local coordinate coding. *NIPS*, 2009.

